# Density Estimation under Independent *Similarly* Distributed Sampling Assumptions

**Tony Jebara, Yingbo Song and Kapil Thadani**
Department of Computer Science
Columbia University
New York, NY 10027
{ jebara,yingbo,kapil }@cs.columbia.edu

## Abstract

A method is proposed for semiparametric estimation where parametric and non-parametric criteria are exploited in density estimation and unsupervised learning. This is accomplished by making sampling assumptions on a dataset that smoothly interpolate between the extreme of independently distributed (or *id*) sample data (as in nonparametric kernel density estimators) to the extreme of independent *identically* distributed (or *iid*) sample data. This article makes independent *similarly* distributed (or *isd*) sampling assumptions and interpolates between these two using a scalar parameter. The parameter controls a Bhattacharyya affinity penalty between pairs of distributions on samples. Surprisingly, the *isd* method maintains certain consistency and unimodality properties akin to maximum likelihood estimation. The proposed *isd* scheme is an alternative for handling nonstationarity in data without making drastic hidden variable assumptions which often make estimation difficult and laden with local optima. Experiments in density estimation on a variety of datasets confirm the value of *isd* over *iid* estimation, *id* estimation and mixture modeling.

## 1  Introduction

Density estimation is a popular unsupervised learning technique for recovering distributions from data. Most approaches can be split into two categories: parametric methods where the functional form of the distribution is known a priori (often from the exponential family (Collins et al., 2002; Efron & Tibshirani, 1996)) and non-parametric approaches which explore a wider range of distributions with less constrained forms (Devroye & Gyorfi, 1985). Parametric approaches can underfit or may be mismatched to real-world data if they are built on incorrect a priori assumptions. A popular non-parametric approach is kernel density estimation or the Parzen windows method (Silverman, 1986). However, these may over-fit thus requiring smoothing, bandwidth estimation and adaptation (Wand & Jones, 1995; Devroye & Gyorfi, 1985; Bengio et al., 2005). Semiparametric efforts (Olking & Spiegelman, 1987) combine the complementary advantages of both schools. For instance, mixture models in their infinite-component setting (Rasmussen, 1999) as well as statistical processes (Teh et al., 2004) make only partial parametric assumptions. Alternatively, one may seed non-parametric distributions with parametric assumptions (Hjort & Glad, 1995) or augment parametric models with nonparametric factors (Naito, 2004). This article instead proposes a continuous interpolation between *iid* parametric density estimation and *id* kernel density estimation. It makes independent *similarly* distributed (*isd*) sampling assumptions on the data. In *isd*, a scalar parameter $\lambda$ trades off parametric and non-parametric properties to produce an overall better density estimate. The method avoids sampling or approximate inference computations and only recycles well known parametric update rules for estimation. It remains computationally efficient, unimodal and consistent for a wide range of models.

This paper is organized as follows. Section 2 shows how *id* and *iid* sampling setups can be smoothly interpolated using a novel *isd* posterior which maintains log-concavity for many popular models. Section 3 gives analytic formulae for the exponential family case as well as slight modifications to familiar maximum likelihood updates for recovering parameters under *isd* assumptions. Some consistency properties of the *isd* posterior are provided. Section 4 then extends the method to hidden variable models or mixtures and provides simple update rules. Section 5 provides experiments comparing *isd* with *id* and *iid* as well as mixture modeling. We conclude with a brief discussion.

## 2 A Continuum between *id* and *iid*

Assume we are given a dataset of $N-1$ inputs $\mathbf{x}_1, \ldots, \mathbf{x}_{N-1}$ from some sample space $\Omega$. Given a new query input $\mathbf{x}_N$ also in the same sample space, density estimation aims at recovering a density function $p(\mathbf{x}_1, \ldots, \mathbf{x}_{N-1}, \mathbf{x}_N)$ or $p(\mathbf{x}_N|\mathbf{x}_1, \ldots, \mathbf{x}_{N-1})$ using a Bayesian or frequentist approach. Therefore, a general density estimation task is, given a dataset $\mathcal{X} = \mathbf{x}_1, \ldots, \mathbf{x}_N$, recover $p(\mathbf{x}_1, \ldots, \mathbf{x}_N)$. A common subsequent assumption is that the data points are *id* or independently sampled which leads to the following simplification:

$$p^{id}(\mathcal{X}) \quad = \quad \prod_{n=1}^{N} p_n(\mathbf{x}_n).$$

The joint likelihood factorizes into a product of independent singleton marginals $p_n(\mathbf{x}_n)$ each of which can be different. A stricter assumption is that all samples share the *same* singleton marginal:

$$p^{iid}(\mathcal{X}) \quad = \quad \prod_{n=1}^{N} p(\mathbf{x}_n).$$

which is the popular *iid* sampling situation. In maximum likelihood estimation, either of the above likelihood scores ($p^{id}$ or $p^{iid}$) is maximized by exploring different settings of the marginals. The *id* setup gives rise to what is commonly referred to as kernel density or Parzen estimation. Meanwhile, the *iid* setup gives rise to traditional *iid* parametric maximum likelihood (ML) or maximum a posteriori (MAP) estimation. Both methods have complementary advantages and disadvantages. The *iid* assumption may be too aggressive for many real world problems. For instance, data may be generated by some slowly time-varying nonstationary distribution or (more distressingly) from a distribution that does not match our parametric assumptions. Similarly, the *id* setup may be too flexible and might over-fit when the marginal $p_n(\mathbf{x})$ is myopically recovered from a single $\mathbf{x}_n$.

Consider the parametric ML and MAP setting where parameters $\Theta = \{\theta_1, \ldots, \theta_N\}$ are used to define the marginals. We will use $p(\mathbf{x}|\theta_n) = p_n(\mathbf{x})$ interchangeably. The MAP *id* parametric setting involves maximizing the following posterior (likelihood times a prior) over the models:

$$p^{id}(\mathcal{X}, \Theta) \quad = \quad \prod_{n=1}^{N} p(\mathbf{x}_n|\theta_n)p(\theta_n).$$

To mimic ML, simply set $p(\theta_n)$ to uniform. For simplicity assume that these singleton priors are always kept uniform. Parameters $\Theta$ are then estimated by maximizing $p^{id}$. To obtain the *iid* setup, we can maximize $p^{id}$ subject to constraints that force all marginals to be equal, in other words $\theta_m = \theta_n$ for all $m, n \in \{1, \ldots, N\}$.

Instead of applying $N(N-1)/2$ hard pairwise constraints in an *iid* setup, consider imposing penalty functions across pairs of marginals. These penalty functions reduce the posterior score when marginals disagree and encourage some *stickiness* between models (Teh et al., 2004). We measure the level of agreement between two marginals $p_m(\mathbf{x})$ and $p_n(\mathbf{x})$ using the following Bhattacharyya affinity metric (Bhattacharyya, 1943) between two distributions:

$$\mathcal{B}(p_m, p_n) \quad = \quad \mathcal{B}(p(\mathbf{x}|\theta_m), p(\mathbf{x}|\theta_n)) \quad = \quad \int p^{\beta}(\mathbf{x}|\theta_m)p^{\beta}(\mathbf{x}|\theta_n)d\mathbf{x}.$$

This is a symmetric non-negative quantity in both distributions $p_m$ and $p_n$. The natural choice for the setting of $\beta$ is $1/2$ and in this case, it is easy to verify the affinity is maximal and equals one if and only if $p_m(\mathbf{x}) = p_n(\mathbf{x})$. A large family of alternative information divergences exist

to relate pairs of distributions (Topsoe, 1999) and are discussed in the Appendix. In this article, the Bhattacharyya affinity is preferred since it has some useful computational, analytic, and log-concavity properties. In addition, it leads to straightforward variants of the estimation algorithms as in the *id* and *iid* situations for many choices of parametric densities. Furthermore, (unlike Kullback Leibler divergence) it is possible to compute the Bhattacharyya affinity analytically and efficiently for a wide range of probability models including hidden Markov models (Jebara et al., 2004).

We next define (up to a constant scaling) the posterior score for independent *similarly* distributed (*isd*) data:

$$p_\lambda(\mathcal{X}, \Theta) \quad \propto \quad \prod_n p(\mathbf{x}_n|\theta_n)p(\theta_n) \prod_{m \neq n} \mathcal{B}^{\lambda/N}(p(\mathbf{x}|\theta_m), p(\mathbf{x}|\theta_n)). \tag{1}$$

Here, a scalar power $\lambda/N$ is applied to each affinity. The parameter $\lambda$ adjusts the importance of the similarity between pairs of marginals. Clearly, if $\lambda \to 0$, then the affinity is always unity and the marginals are completely unconstrained as in the *id* setup. Meanwhile, as $\lambda \to \infty$, the affinity is zero unless the marginals are exactly identical. This produces the *iid* setup. We will refer to the *isd* posterior as Equation 1 and when $p(\theta_n)$ is set to uniform, we will call it the *isd* likelihood. One can also view the additional term in *isd* as *id* estimation with a *modified* prior $\tilde{p}(\Theta)$ as follows:

$$\tilde{p}(\Theta) \quad \propto \quad \prod_n p(\theta_n) \prod_{m \neq n} \mathcal{B}^{\lambda/N}(p(\mathbf{x}|\theta_m), p(\mathbf{x}|\theta_n)).$$

This prior is a Markov random field tying all parameters in a pairwise manner in addition to the standard singleton potentials in the *id* scenario. However, this perspective is less appealing since it disguises the fact that the samples are not quite *id* or *iid*.

One of the appealing properties of *iid* and *id* maximum likelihood estimation is its unimodality for log-concave distributions. The *isd* posterior also benefits from a unique optimum and log-concavity. However, the conditional distributions $p(\mathbf{x}|\theta_n)$ are required to be *jointly* log-concave in both parameters $\theta_n$ and data $\mathbf{x}$. This set of distributions includes the Gaussian distribution (with fixed variance) and many exponential family distributions such as the Poisson, multinomial and exponential distribution. We next show that the *isd* posterior score for log-concave distributions is log-concave in $\Theta$. This produces a unique estimate for the parameters as was the case for *id* and *iid* setups.

**Theorem 1** *The* isd *posterior is log-concave for jointly log-concave density distributions and for log-concave prior distributions.*

**Proof 1** *The* isd *log-posterior is the sum of the* id *log-likelihoods, the singleton log-priors and pairwise log-Bhattacharyya affinities:*

$$\log p_\lambda(\mathcal{X}, \Theta) \quad = \quad const + \sum_n \log p(\mathbf{x}_n|\theta_n) + \sum_n \log p(\theta_n) + \frac{\lambda}{N} \sum_n \sum_{m \neq n} \log \mathcal{B}(p_m, p_n).$$

*The* id *log-likelihood is the sum of the log-probabilities of distributions that are log-concave in the parameters and is therefore concave. Adding the log-priors maintains concavity since these are log-concave in the parameters. The Bhattacharyya affinities are log-concave by the following key result (Prekopa, 1973). The Bhattacharyya affinity for log-concave distributions is given by the integral over the sample space of the product of two distributions. Since the term in the integral is a product of jointly log-concave distributions (by assumption), the integrand is a jointly log-concave function. Integrating a log-concave function over some of its arguments produces a log-concave function in the remaining arguments (Prekopa, 1973). Therefore, the Bhattacharyya affinity is log-concave in the parameters of jointly log-concave distributions. Finally, since the* isd *log-posterior is the sum of concave terms and concave log-Bhattacharyya affinities, it must be concave.*

This log-concavity permits iterative and greedy maximization methods to reliably converge in practice. Furthermore, the *isd* setup will produce convenient update rules that build upon *iid* estimation algorithms. There are additional properties of *isd* which are detailed in the following sections. We first explore the $\beta = 1/2$ setting and subsequently discuss the $\beta = 1$ setting.

# 3 Exponential Family Distributions and $\beta = 1/2$

We first specialize the above derivations to the case where the singleton marginals obey the *exponential family* form as follows:

$$p(\mathbf{x}|\theta_n) \quad = \quad \exp\left(H(\mathbf{x}) + \theta_n^\mathsf{T} T(\mathbf{x}) - A(\theta_n)\right).$$

An exponential family distribution is specified by providing $H$, the Lebesgue-Stieltjes integrator, $\theta_n$ the vector of natural parameters, $T$, the sufficient statistic, and $A$ the normalization factor (which is also known as the cumulant-generating function or the log-partition function). Tables of these values are shown in (Jebara et al., 2004). The function $A$ is obtained by normalization (a Legendre transform) and is convex by construction. Therefore, exponential family distributions are always log-concave in the parameters $\theta_n$. For the exponential family, the Bhattacharyya affinity is computable in closed form as follows:

$$\mathcal{B}(p_m, p_n) \quad = \quad \exp\left(A(\theta_m/2 + \theta_n/2) - A(\theta_m)/2 - A(\theta_n)/2\right).$$

Assuming uniform priors on the exponential family parameters, it is now straightforward to write an iterative algorithm to maximize the *isd* posterior. We find settings of $\theta_1, \ldots, \theta_N$ that maximize the *isd* posterior or $\log p_\lambda(\mathcal{X}, \Theta)$ using a simple greedy method. Assume a current set of parameters is available $\tilde{\theta}_1, \ldots, \tilde{\theta}_N$. We then update a single $\theta_n$ to increase the posterior while all other parameters (denoted $\tilde{\Theta}_{/n}$) remain fixed at their previous settings. It suffices to consider only terms in $\log p_\lambda(\mathcal{X}, \Theta)$ that are variable with $\theta_n$:

$$\log p_\lambda(\mathcal{X}, \theta_n, \tilde{\Theta}_{/n}) \quad = \quad const + \theta_n^\mathsf{T} T(\mathbf{x}_n) - \frac{N + \lambda(N-1)}{N} A(\theta_n) + \frac{2\lambda}{N} \sum_{m \neq n} A(\tilde{\theta}_m/2 + \theta_n/2).$$

If the exponential family is *jointly* log-concave in parameters and data (as is the case for Gaussians), this term is log-concave in $\theta_n$. Therefore, we can take a partial derivative of it with respect to $\theta_n$ and set to zero to maximize:

$$A'(\theta_n) \quad = \quad \frac{N}{N + \lambda(N-1)} \left(T(\mathbf{x}_n) + \frac{\lambda}{N} \sum_{m \neq n} A'(\tilde{\theta}_m/2 + \theta_n/2)\right). \tag{2}$$

For the Gaussian mean case (i.e. a white Gaussian with covariance locked at identity), we have $A(\theta) = \theta^T \theta$. Then a closed-form formula is easy to recover from the above[1]. However, a simpler iterative update rule for $\theta_n$ is also possible as follows. Since $A(\theta)$ is a convex function, we can compute a linear variational lower bound on each $A(\theta_m/2 + \theta_n/2)$ term for the current setting of $\theta_n$:

$$\begin{aligned} \log p_\lambda(\mathcal{X}, \theta_n, \tilde{\Theta}_{/n}) \quad \geq \quad & const + \theta_n^\mathsf{T} T(\mathbf{x}_n) - \frac{N + \lambda(N-1)}{N} A(\theta_n) \\ & + \frac{\lambda}{N} \sum_{m \neq n} 2A(\tilde{\theta}_m/2 + \tilde{\theta}_n/2) + A'(\tilde{\theta}_m/2 + \tilde{\theta}_n/2)^\mathsf{T}(\theta_n - \tilde{\theta}_n). \end{aligned}$$

This gives an iterative update rule of the form of Equation 2 where the $\theta_n$ on the right hand side is kept fixed at its previous setting (i.e. replace the right hand side $\theta_n$ with $\tilde{\theta}_n$) while the equation is iterated multiple times until the value of $\theta_n$ converges. Since we have a variational lower bound, each iterative update of $\theta_n$ monotonically increases the *isd* posterior. We can also work with a robust (yet not log-concave) version of the *isd* score which has the form:

$$\log \hat{p}_\lambda(\mathcal{X}, \Theta) \quad = \quad const + \sum_n \log p(\mathbf{x}_n|\theta_n) + \sum_n \log p(\theta_n) + \frac{\lambda}{N} \sum_n \log\left(\sum_{m \neq n} \mathcal{B}(p_m, p_n)\right).$$

and leads to the general update rule (where $\alpha = 0$ reproduces *isd* and larger $\alpha$ increases robustness):

$$A'(\theta_n) = \frac{N}{N + \lambda(N-1)} \left(T(\mathbf{x}_n) + \frac{\lambda}{N} \sum_{m \neq n} \frac{(N-1)\mathcal{B}^\alpha(p(\mathbf{x}|\tilde{\theta}_m), p(\mathbf{x}|\tilde{\theta}_n))}{\sum_{l \neq n} \mathcal{B}^\alpha(p(\mathbf{x}|\tilde{\theta}_l), p(\mathbf{x}|\tilde{\theta}_n))} A'(\tilde{\theta}_m/2 + \tilde{\theta}_n/2)\right).$$

We next examine marginal consistency, another important property of the *isd* posterior.

## 3.1 Marginal Consistency in the Gaussian Mean Case

For marginal consistency, if a datum and model parameter are hidden and integrated over, this should not change our estimate. It is possible to show that the *isd* posterior is marginally consistent at least in the Gaussian mean case (one element of the exponential family). In other words, marginalizing over an observation and its associated marginal's parameter (which can be taken to be $\mathbf{x}_N$ and $\theta_N$ without loss of generality) still produces a similar *isd* posterior on the remaining observations $\mathcal{X}_{/N}$ and parameters $\Theta_{/N}$. Thus, we need:

$$\int \int p_\lambda(\mathcal{X}, \Theta) d\mathbf{x}_N d\theta_N \quad \propto \quad p_\lambda(\mathcal{X}_{/N}, \Theta_{/N}).$$

We then would recover the posterior formed using the formula in Equation 1 with only $N - 1$ observations and $N - 1$ models.

**Theorem 2** *The* isd *posterior with* $\beta = 1/2$ *is marginally consistent for Gaussian distributions.*

**Proof 2** *Start by integrating over* $\mathbf{x}_N$:

$$\int p_\lambda(\mathcal{X}, \Theta) d\mathbf{x}_N \quad \propto \quad \prod_{i=1}^{N-1} p(\mathbf{x}_i|\theta_i) \prod_{n=1}^{N} p(\theta_n) \prod_{m=n+1}^{N} \mathcal{B}^{2\lambda/N}(p_m, p_n)$$

*Assume the singleton prior* $p(\theta_N)$ *is uniform and integrate over* $\theta_N$ *to obtain:*

$$\int \int p_\lambda(\mathcal{X}, \Theta) d\mathbf{x}_N d\theta_N \quad \propto \quad \prod_{i=1}^{N-1} p(\mathbf{x}_i|\theta_i) \prod_{n=1}^{N-1} \prod_{m=n+1}^{N-1} \mathcal{B}^{2\lambda/N}(p_m, p_n) \int \prod_{m=1}^{N-1} \mathcal{B}^{2\lambda/N}(p_m, p_N) d\theta_N$$

*Consider only the right hand integral and impute the formula for the Bhattacharyya affinity:*

$$\int \prod_{m=1}^{N-1} \mathcal{B}^{2\lambda/N}(p_m, p_N) d\theta_N \quad = \quad \int \exp\left( \frac{2\lambda}{N} \sum_{m=1}^{N-1} A\left(\frac{\theta_m}{2} + \frac{\theta_N}{2}\right) - \frac{A(\theta_m)}{2} - \frac{A(\theta_N)}{2} \right) d\theta_N$$

*In the (white) Gaussian case* $A(\theta) = \theta^T \theta$ *which simplifies the above into:*

$$\int \prod_{m=1}^{N-1} \mathcal{B}^{2\lambda/N}(p_m, p_N) d\theta_N \quad = \quad \int \exp\left( -\frac{2\lambda}{N} \sum_{m=1}^{N-1} A\left(\frac{\theta_m}{2} - \frac{\theta_N}{2}\right) \right) d\theta_N$$

$$\propto \quad \exp\left( \frac{2\lambda}{N(N-1)} \sum_{n=1}^{N-1} \sum_{m=n+1}^{N-1} A\left(\frac{\theta_m}{2} + \frac{\theta_n}{2}\right) - \frac{A(\theta_m)}{2} - \frac{A(\theta_n)}{2} \right)$$

$$\propto \quad \prod_{n=1}^{N-1} \prod_{m=n+1}^{N-1} \mathcal{B}^{\frac{2\lambda}{N(N-1)}}(p_m, p_n)$$

*Reinserting the integral changes the exponent of the pairs of Bhattacharyya affinities between the* $(N-1)$ *models raising it to the appropriate power* $\lambda/(N-1)$:

$$\int \int p_\lambda(\mathcal{X}, \Theta) d\mathbf{x}_N d\theta_N \quad \propto \quad \prod_{i=1}^{N-1} p(\mathbf{x}_i|\theta_i) \prod_{n=1}^{N-1} \prod_{m=n+1}^{N-1} \mathcal{B}^{2\lambda/(N-1)}(p_m, p_n) = p_\lambda(\mathcal{X}_{/N}, \Theta_{/N}).$$

Therefore, we get the same *isd* score that we would have obtained had we started with only $(N-1)$ data points. We conjecture that it is possible to generalize the marginal consistency argument to other distributions beyond the Gaussian. The *isd* estimator thus has useful properties and still agrees with *id* when $\lambda = 0$ and *iid* when $\lambda = \infty$. Next, the estimator is generalized to handle distributions beyond the exponential family where latent variables are implicated (as is the case for mixtures of Gaussians, hidden Markov models, latent graphical models and so on).

## 4 Hidden Variable Models and $\beta = 1$

One important limitation of most divergences between distributions is that they become awkward when dealing with hidden variables or mixture models. This is because they may involve intractable integrals. The Bhattacharyya affinity with the setting $\beta = 1$, also known as the probability product kernel, is an exception to this since it only involves integrating the product of two distributions. In fact, it is known that this affinity is efficient to compute for mixtures of Gaussians, multinomials and even hidden Markov models (Jebara et al., 2004). This permits the affinity metric to efficiently pull together parameters $\theta_m$ and $\theta_n$. However, for mixture models, there is the presence of hidden variables $\mathbf{h}$ in addition to observed variables. Therefore, we replace all the marginals $p(\mathbf{x}|\theta_n) = \sum_{\mathbf{h}} p(\mathbf{x}, \mathbf{h}|\theta_n)$. The affinity is still straightforward to compute for any pair of latent variable models (mixture models, hidden Markov models and so on). Thus, evaluating the *isd* posterior is straightforward for such models when $\beta = 1$. We next provide a variational method that makes it possible to maximize a lower bound on the *isd* posterior in these cases.

Assume a current set of parameters is available $\tilde{\Theta} = \tilde{\theta}_1, \ldots, \tilde{\theta}_N$. We will find a new setting for $\theta_n$ that increases the posterior while all other parameters (denoted $\tilde{\Theta}_{/n}$) remain fixed at their previous settings. It suffices to consider only terms in $\log p_\lambda(\mathcal{X}, \Theta)$ that depend on $\theta_n$. This yields:

$$
\begin{aligned}
\log p_\lambda(\mathcal{X}, \theta_n, \tilde{\Theta}_{/n}) &= const + \log p(\mathbf{x}_n|\theta_n)p(\theta_n) + \frac{2\lambda}{N} \sum_{m \neq n} \log \int p(\mathbf{x}|\tilde{\theta}_m)p(\mathbf{x}|\theta_n)d\mathbf{x} \\
&\geq const + \log p(\mathbf{x}_n|\theta_n)p(\theta_n) + \frac{2\lambda}{N} \sum_{m \neq n} \int p(\mathbf{x}|\tilde{\theta}_m) \log p(\mathbf{x}|\theta_n)d\mathbf{x}
\end{aligned}
$$

The application of Jensen's inequality above produces an auxiliary function $\mathcal{Q}(\theta_n|\tilde{\Theta}_{/n})$ which is a lower-bound on the log-posterior. Note that each density function has hidden variables, $p(\mathbf{x}_n|\theta_n) = \sum_{\mathbf{h}} p(\mathbf{x}_n, \mathbf{h}|\theta_n)$. Applying Jensen's inequality again (as in the Expectation-Maximization or EM algorithm) replaces the log-incomplete likelihoods over $\mathbf{h}$ with expectations over the complete posteriors given the previous parameters $\tilde{\theta}_n$. This gives *isd* the following auxiliary function $\mathcal{Q}(\theta_n|\tilde{\Theta}) =$

$$
\sum_{\mathbf{h}} p(\mathbf{h}|\mathbf{x}_n, \tilde{\theta}_n) \log p(\mathbf{x}_n, \mathbf{h}|\theta_n) + \log p(\theta_n) + \frac{2\lambda}{N} \sum_{m \neq n} \int p(\mathbf{x}|\tilde{\theta}_m) \sum_{\mathbf{h}} p(\mathbf{h}|\mathbf{x}, \tilde{\theta}_n) \log p(\mathbf{x}, \mathbf{h}|\theta_n)d\mathbf{x}.
$$

This is a variational lower bound which can be iteratively maximized instead of the original *isd* posterior. While it is possible to directly solve for the maximum of $\mathcal{Q}(\theta_n|\tilde{\Theta})$ in some mixture models, in practice, a further simplification is to replace the integral over $\mathbf{x}$ with synthesized samples drawn from $p(\mathbf{x}|\tilde{\theta}_m)$. This leads to the following approximate auxiliary function (based on the law of large numbers) which is merely the update rule for EM for $\theta_n$ with $s = 1, \ldots, S$ virtual samples $\mathbf{x}_{m,s}$ obtained from the $m$'th model $p(\mathbf{x}|\tilde{\theta}_m)$ for each of the other $N - 1$ models, $\tilde{\mathcal{Q}}(\theta_n|\tilde{\Theta}) =$

$$
\sum_{\mathbf{h}} p(\mathbf{h}|\mathbf{x}_n, \tilde{\theta}_n) \log p(\mathbf{x}_n, \mathbf{h}|\theta_n) + \log p(\theta_n) + \frac{2\lambda}{SN} \sum_{m \neq n} \sum_s \sum_{\mathbf{h}} p(\mathbf{h}|\mathbf{x}_{m,s}, \tilde{\theta}_n) \log p(\mathbf{x}_{m,s}, \mathbf{h}|\theta_n).
$$

We now have an efficient update rule for latent variable models (mixtures, hidden Markov models, etc.) which maximizes a lower bound on $p_\lambda(\mathcal{X}, \Theta)$. Unfortunately, as with most EM implementations, the arguments for log-concavity no longer hold.

## 5 Experiments

A preliminary way to evaluate the usefulness of the *isd* framework is to explore density estimation over real-world datasets under varying $\lambda$. If we set $\lambda$ large, we have the standard *iid* setup and only fit a single parametric model to the dataset. For small $\lambda$, we obtain the kernel density or Parzen estimator. In between, an iterative algorithm is available to maximize the *isd* posterior to obtain potentially superior models $\theta_1^*, \ldots, \theta_N^*$. Figure 1 shows the *isd* estimator with Gaussian models on a ring-shaped 2D dataset. The new estimator recovers the shape of the distribution more accurately. To evaluate performance on real data, we aggregate the *isd* learned models into a single density estimate as is done with Parzen estimators and compute the *iid* likelihood of held out test

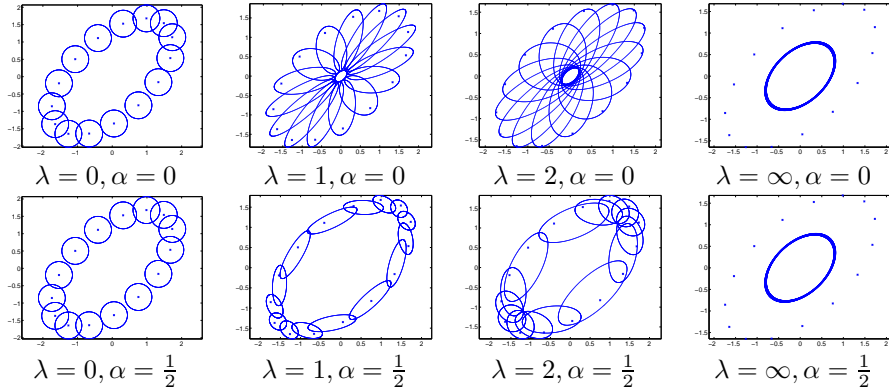

Figure 1: Estimation with *isd* for Gaussian models (mean and covariance) on synthetic data.

| Dataset | *id* | *iid*-1 | *iid*-2 | *iid*-3 | *iid*-4 | *iid*-5 | *iid*-$\infty$ | *isd* $\alpha = 0$ | *isd* $\alpha = \frac{1}{2}$ |
|---|---|---|---|---|---|---|---|---|---|
| SPIRAL | -5.61e3 | -1.36e3 | -1.36e3 | -1.19e3 | -7.98e2 | -6.48e2 | -4.86e2 | -2.26e2 | **-1.19e2** |
| MIT-CBCL | -9.82e2 | -1.39e3 | -1.19e3 | -1.00e3 | -1.01e3 | -1.10e3 | -3.14e3 | **-9.79e2** | **-9.79e2** |
| HEART | -1.94e3 | -2.02e4 | -3.23e4 | -2.50e4 | -1.68e4 | -3.15e4 | **-4.02e2** | -4.51e2 | -4.47e2 |
| DIABETES | -6.25e3 | -2.12e5 | -2.85e5 | -4.48e5 | -2.03e5 | -3.40e5 | -8.22e2 | -8.28e2 | **-8.09e2** |
| CANCER | -5.80e3 | -7.22e6 | -2.94e6 | -3.92e6 | -4.08e6 | -3.96e6 | **-1.22e2** | -5.54e2 | -5.54e2 |
| LIVER | -3.41e3 | -2.53e4 | -1.88e4 | -2.79e4 | -2.62e4 | -3.23e4 | **-4.56e2** | -4.74e2 | -4.69e2 |

Table 1: Gaussian test log-likelihoods using *id*, *iid*, EM, $\infty$ GMM and *isd* estimation.

data via $\sum_\tau \log\left(\frac{1}{N}\sum_n p(\mathbf{x}_\tau|\theta_n^*)\right)$. A larger score implies a better $p(\mathbf{x})$ density estimate. Table 1 summarizes experiments with the Gaussian (mean and covariance) models. On 6 standard datasets, we show the average test log-likelihood of Gaussian estimation while varying the settings of $\lambda$ compared to a single *iid* Gaussian, an *id* Parzen RBF estimator and a mixture of 2 to 5 Gaussians using EM. Comparisons with (Rasmussen, 1999) are also shown. Cross-validation was used to choose the $\sigma$, $\lambda$ or EM local minimum (from ten initializations), for the *id*, *isd* and EM algorithms respectively. Train, cross-validation and test split sizes where 80%, 10% and 10% respectively. The test log-likelihoods show that *isd* outperformed *iid*, *id* and EM estimation and was comparable to infinite Gaussian mixture ($iid-\infty$) models (Rasmussen, 1999) (which is a far more computationally demanding method). In another synthetic experiment with hidden Markov models, 40 sequences of 8 binary symbols were generated using 2 state HMMs with 2 discrete emissions. However, the parameters generating the HMMs were allowed to slowly drift during sampling (i.e. not *iid*). The data was split into 20 training and 20 testing examples. Table 2 shows that the *isd* estimator for certain values of $\lambda$ produced higher test log-likelihoods than *id* and *iid*.

## 6  Discussion

This article has provided an *isd* scheme to smoothly interpolate between *id* and *iid* assumptions in density estimation. This is done by penalizing divergence between pairs of models using a Bhattacharyya affinity. The method maintains simple update rules for recovering parameters for exponential families as well as mixture models. In addition, the *isd* posterior maintains useful log-concavity and marginal consistency properties. Experiments show its advantages in real-world datasets where *id* or *iid* assumptions may be too extreme. Future work involves extending the approach into other aspects of unsupervised learning such as clustering. We are also considering computing the *isd* pos-

| $\lambda = 0$ | $\lambda = 1$ | $\lambda = 2$ | $\lambda = 3$ | $\lambda = 4$ | $\lambda = 5$ | $\lambda = 10$ | $\lambda = 20$ | $\lambda = 30$ | $\lambda = \infty$ |
|---|---|---|---|---|---|---|---|---|---|
| -5.7153 | -5.5875 | -5.5692 | **-5.5648** | -5.5757 | -5.5825 | -5.5849 | -5.5856 | -5.6152 | -5.5721 |

Table 2: HMM test log-likelihoods using *id*, *iid* and *isd* estimation.

terior with a normalizing constant which depends on $\lambda$ and thus permits a direct estimate of $\lambda$ by maximization instead of cross-validation[2].

# 7 Appendix: Alternative Information Divergences

There is a large family of information divergences (Topsoe, 1999) between pairs of distributions (Renyi measure, variational distance, $\chi^2$ divergence, etc.) that can be used to pull models $p_m$ and $p_n$ towards each other. The Bhattacharya, though, is computationally easier to evaluate and minimize over a wide range of probability models (exponential families, mixtures and hidden Markov models). An alternative is the Kullback-Leibler divergence $D(p_m\|p_n) = \int p_m(\mathbf{x})(\log p_m(\mathbf{x}) - \log p_n(\mathbf{x}))d\mathbf{x}$ and its symmetrized variant $D(p_m\|p_n)/2 + D(p_n\|p_m)/2$. The Bhattacharyya affinity is related to the symmetrized variant of KL. Consider a variational distribution $q$ that lies between the input $p_m$ and $p_n$. The log Bhattacharyya affinity with $\beta = 1/2$ can be written as follows:

$$\log \mathcal{B}(p_m, p_n) = \log \int q(\mathbf{x}) \frac{\sqrt{p_m(\mathbf{x})p_n(\mathbf{x})}}{q(\mathbf{x})} d\mathbf{x} \geq -D(q\|p_m)/2 - D(q\|p_n)/2.$$

Thus, $\mathcal{B}(p_m, p_n) \geq \exp(-D(q\|p_m)/2 - D(q\|p_n)/2)$. The choice of $q$ that maximizes the lower bound on the Bhattacharyya is $q(\mathbf{x}) = \frac{1}{Z}\sqrt{p_m(\mathbf{x})p_n(\mathbf{x})}$. Here, $Z = \mathcal{B}(p_m, p_n)$ normalizes $q(\mathbf{x})$ and is therefore equal to the Bhattacharyya affinity. Thus we have the following property:

$$-2\log\mathcal{B}(p_m, p_n) = \min_q D(q\|p_m) + D(q\|p_n).$$

It is interesting to note that the Jensen-Shannon divergence (another symmetrized variant of KL) emerges by placing the variational $q$ distribution as the second argument in the divergences:

$$2JS(p_m, p_n) = D(p_m\|p_m/2 + p_n/2) + D(p_n\|p_m/2 + p_n/2) = \min_q D(p_m\|q) + D(p_n\|q).$$

Simple manipulations then show $2JS(p_m, p_n) \leq \min(D(p_m\|p_n), D(p_n\|p_m))$. Thus, there are close ties between Bhattacharyya, Jensen-Shannon and symmetrized KL divergences.

## Footnotes

[1]The update for the Gaussian mean with covariance=$I$ is: $\theta_n = \frac{1}{N + \lambda(N-1)/2}(N\mathbf{x}_n + \lambda/2 \sum_{m \neq n} \tilde{\theta}_m)$.

[2]Work supported in part by NSF Award IIS-0347499 and ONR Award N000140710507.

# References

Bengio, Y., Larochelle, H., & Vincent, P. (2005). Non-local manifold Parzen windows. *Neural Information Processing Systems*.

Bhattacharyya, A. (1943). On a measure of divergence between two statistical populations defined by their probability distributions. *Bull. Calcutta Math Soc.*

Collins, M., Dasgupta, S., & Schapire, R. (2002). A generalization of principal components analysis to the exponential family. *NIPS*.

Devroye, L., & Gyorfi, L. (1985). *Nonparametric density estimation: The $l_1$ view*. John Wiley.

Efron, B., & Tibshirani, R. (1996). Using specially designed exponential families for density estimation. *The Annals of Statistics*, *24*, 2431–2461.

Hjort, N., & Glad, I. (1995). Nonparametric density estimation with a parametric start. *The Annals of Statistics*, *23*, 882–904.

Jebara, T., Kondor, R., & Howard, A. (2004). Probability product kernels. *Journal of Machine Learning Research*, *5*, 819–844.

Naito, K. (2004). Semiparametric density estimation by local $l_2$-fitting. *The Annals of Statistics*, *32*, 1162–1192.

Olking, I., & Spiegelman, C. (1987). A semiparametric approach to density estimation. *Journal of the American Statistcal Association*, *82*, 858–865.

Prekopa, A. (1973). On logarithmic concave measures and functions. *Acta. Sci. Math.*, *34*, 335–343.

Rasmussen, C. (1999). The infinite Gaussian mixture model. *NIPS*.

Silverman, B. (1986). *Density estimation for statistics and data analysis*. Chapman and Hall: London.

Teh, Y., Jordan, M., Beal, M., & Blei, D. (2004). Hierarchical Dirichlet processes. *NIPS*.

Topsoe, F. (1999). Some inequalities for information divergence and related measures of discrimination. *Journal of Inequalities in Pure and Applied Mathematics*, *2*.

Wand, M., & Jones, M. (1995). *Kernel smoothing*. CRC Press.

